# A Unified Gradient-Descent/Clustering Architecture for Finite State Machine Induction

Sreerupa Das and Michael C. Mozer
Department of Computer Science
University of Colorado
Boulder, CO 80309–0430

## Abstract

Although recurrent neural nets have been moderately successful in learning to emulate finite-state machines (FSMs), the continuous internal state dynamics of a neural net are not well matched to the discrete behavior of an FSM. We describe an architecture, called DOLCE, that allows discrete states to evolve in a net as learning progresses. DOLCE consists of a standard recurrent neural net trained by gradient descent and an adaptive clustering technique that quantizes the state space. DOLCE is based on the assumption that a finite set of discrete internal states is required for the task, and that the actual network state belongs to this set but has been corrupted by noise due to inaccuracy in the weights. DOLCE learns to recover the discrete state with maximum a posteriori probability from the noisy state. Simulations show that DOLCE leads to a significant improvement in generalization performance over earlier neural net approaches to FSM induction.

## 1 INTRODUCTION

Researchers often try to understand—post hoc—representations that emerge in the hidden layers of a neural net following training. Interpretation is difficult because these representations are typically highly distributed and *continuous*. By "continuous," we mean that if one constructed a scatterplot over the hidden unit activity space of patterns obtained in response to various inputs, examination at any scale would reveal the patterns to be broadly distributed over the space.

Continuous representations aren't always appropriate. Many task domains seem to require *discrete* representations—representations selected from a finite set of alternatives. If a neural net learned a discrete representation, the scatterplot over hidden activity space would show points to be superimposed at fine scales of analysis. Some

examples of domains in which discrete representations might be desirable include: finite-state machine emulation, data compression, language and higher cognition (involving discrete symbol processing), and categorization in the context of decision making. In such domains, standard neural net learning procedures, which have a propensity to produce continuous representations, may not be appropriate. The work we report here involves designing an inductive bias into the learning procedure in order to encourage the formation of discrete internal representations.

In the recent years, various approaches have been explored for learning discrete representations using neural networks (McMillan, Mozer, & Smolensky, 1992; Mozer & Bachrach, 1990; Mozer & Das, 1993; Schütze, 1993; Towell & Shavlik, 1992). However, these approaches are domain specific, making strong assumptions about the nature of the task. In our work, we describe a general methodology that makes no assumption about the domain to which it is applied, beyond the fact that discrete representations are desireable.

## 2   FINITE STATE MACHINE INDUCTION

We illustrate the methodology using the domain of finite-state machine (FSM) induction. An FSM defines a class of symbol strings. For example, the class $(10)^*$ consists of all strings with one or more repetitions of 10; 101010 is a positive example of the class, 111 is a negative example. An FSM consists principally of a finite set of states and a function that maps the current state and the current symbol of the string into a new state. Certain states of the FSM are designated "accept" states, meaning that if the FSM ends up in these states, the string is a member of the class. The induction problem is to infer an FSM that parsimoniously characterizes the positive and negative exemplars, and hence characterizes the underlying class.

A generic recurrent net architecture that could be used for FSM emulation and induction is shown on the left side of Figure 1. A string is presented to the input layer of the net, one symbol at a time. Following the end of the string, the net should output whether or not the string is a member of the class. The hidden unit activity pattern at any point during presentation of a string corresponds to the internal state of an FSM.

Such a net, trained by a gradient descent procedure, is able to learn to perform this or related tasks (Elman, 1990; Giles et al., 1992; Pollack, 1991; Servan-Schreiber, Cleeremans, & McClelland, 1991; Watrous & Kuhn, 1992). Although these models have been relatively successful in learning to emulate FSMs, the continuous internal state dynamics of a neural net are not well matched to the discrete behavior of FSMs. Roughly, regions of hidden unit activity space can be identified with states in an FSM, but because the activities are continuous, one often observes the network drifting from one state to another. This occurs especially with input strings longer than those on which the network was trained.

To achieve more robust dynamics, one might consider quantizing the hidden state. Two approaches to quantization have been explored previously. In the first, a net is trained in the manner described above. After training, the hidden state space is partitioned into disjoint regions and each hidden activity pattern is then discretized by mapping it to the center of its corresponding region (Das & Das, 1991; Giles

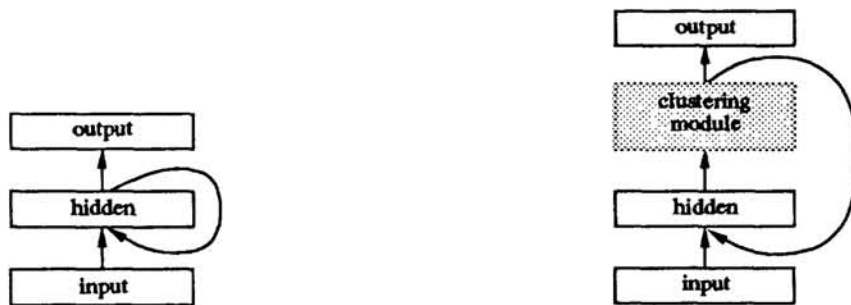

Figure 1: On the left is a generic recurrent architecture that could be used for FSM induction. Each box corresponds to a layer of units, and arrows depict complete connectivity between layers. At each time step, a new symbol is presented on the input and the input and hidden representations are integrated to form a new hidden representation. On the right is the general architecture of DOLCE.

et al., 1992). In a second approach, quantization is enforced *during* training by mapping the the hidden state at each time step to the nearest corner of a $[0, 1]^n$ hypercube (Zeng, Goodman, & Smyth, 1993).

Each of these approaches has its limitations. In the first approach, because learning does not consider the latter quantization, the hidden activity patterns that result from learning may not lie in natural clusters. Consequently, the quantization step may not group together activity patterns that correspond to the same state. In the second approach, the quantization process causes the error surface to have discontinuities and to be flat in local neighborhoods of the weight space. Hence, gradient descent learning algorithms cannot be used; instead, even more heuristic approaches are required. To overcome the limitations of these approaches, we have pursued an approach in which quantization is an integral part of the learning process.

## 3  DOLCE

Our approach incorporates a *clustering module* into the recurrent net architecture, as shown on the right side of Figure 1. The hidden layer activities are processed by the clustering module before being passed on to other layers. The clustering module maps regions in hidden state space to a single point in the same space, effectively partitioning or clustering the hidden state space. Each cluster corresponds to a discrete internal state. The clusters are adaptive and dynamic, changing over the course of learning. We call this architecture DOLCE, for dynamic on-line clustering and state extraction.

The DOLCE architecture may be explored along two dimensions: (1) the clustering algorithm used (e.g., a Gaussian mixture model, ISODATA, the Forgy algorithm, vector quantization schemes), and (2) whether supervised or unsupervised training is used to identify the clusters. In unsupervised mode, the performance error on the FSM induction task has no effect on the operation of the clustering algorithm; instead, an internal criterion characterizes goodness of clusters. In supervised mode, the primary measure that affects the goodness of a cluster is the performance error. Regardless of the training mode, all clustering algorithms incorporate a pressure to

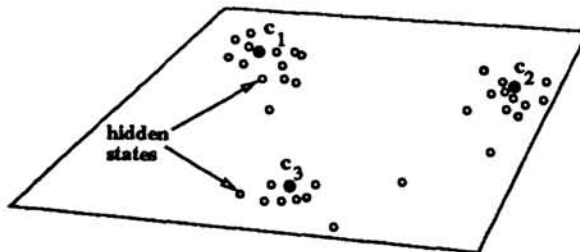

Figure 2: Two dimensions of a typical state space. The true states needed to perform the task are $c_1$, $c_2$, and $c_3$, while the observed hidden states, assumed to be corrupted by noise, are distributed about the $c_i$.

produce a small number of clusters. Additionally, as we elaborate more specifically below, the algorithms must allow for a soft or continuous clustering during training, in order to be integrated into a gradient-based learning procedure.

We have explored two possibilities for the clustering module. The first involves the use of Forgy's algorithm in an unsupervised mode. Forgy's (1965) algorithm determines both the number of clusters and the partitioning of the space. The second uses a Gaussian mixture model in a supervised mode, where the mixture model parameters are adjusted so as to minimize the performance error. Both approaches were successful, but as the latter approach obtained better results, we describe it in the next section.

## 4    CLUSTERING USING A MIXTURE MODEL

Here we motivate the incorporation of a Gaussian mixture model into DOLCE, using an argument that gives the approach a solid theoretical foundation. Several assumptions underly the approach. First, we assume that the task faced by DOLCE is such that it requires a finite set of internal or *true* states, $C = \{c_1, c_2, \ldots, c_T\}$. This is simply the premise that motivates this line of work. Second, we assume that any observed hidden state—i.e., a hidden activity pattern that results from presentation of a symbol sequence—belongs to $C$ but has been corrupted by noise due to inaccuracy in the network weights. Third, we assume that this noise is Gaussian and decreases as learning progresses (i.e., as the weights are adjusted to better perform the task). These assumptions are depicted in Figure 2.

Based on these assumptions, we construct a Gaussian mixture distribution that models the observed hidden states:

$$p(\mathbf{h}|\mathbf{C}, \sigma, \mathbf{q}) = \sum_{i=1}^{T} \frac{q_i}{(2\pi\sigma_i^2)^{H/2}} e^{-|\mathbf{h}-\mathbf{c}_i|^2/2\sigma_i^2}$$

where $\mathbf{h}$ denotes an observed hidden state, $\sigma_i^2$ the variance of the noise that corrupts state $c_i$, $q_i$ is the prior probability that the true state is $c_i$, and $H$ is the dimensionality of the hidden state space. For pedagogical purposes, assume for the time being that the parameters of the mixture distribution—$T$, $\mathbf{C}$, $\sigma$, and $\mathbf{q}$—are all known; in a later section we discuss how these parameters are determined.

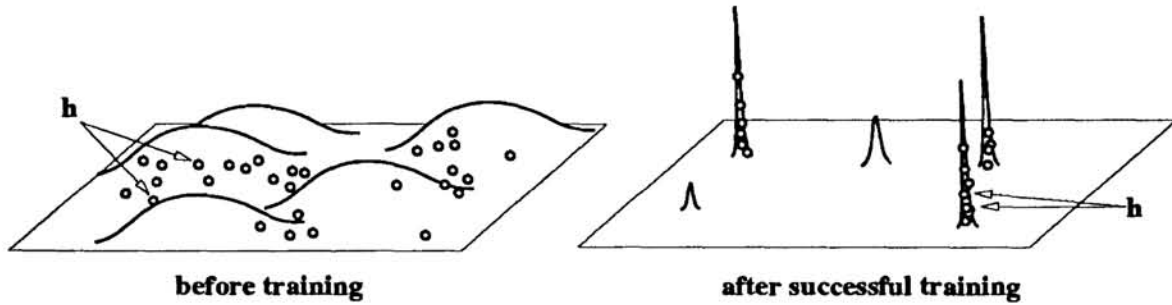

<center>**before training**          **after successful training**</center>

Figure 3: A schematic depiction of the hidden state space before and after training. The horizontal plane represents the state space. The bumps indicate the probability density under the mixture model. Observed hidden states are represented by small open circles.

Given a noisy observed hidden state, $\mathbf{h}$, DOLCE computes the maximum a posteriori (MAP) estimator of $\mathbf{h}$ in $\mathbf{C}$. This estimator then replaces the noisy state and is used in all subsequent computation. The MAP estimator, $\hat{\mathbf{h}}$, is computed as follows. The probability of an observed state $\mathbf{h}$ being generated by a given true state $i$ is

$$p(\mathbf{h}|\text{true state } i) = (2\pi\sigma_i^2)^{-\frac{H}{2}} e^{-|\mathbf{h}-\mathbf{c}_i|^2/2\sigma_i^2}.$$

Using Bayes' rule, one can compute the posterior probability of true state $i$, given that $\mathbf{h}$ has been observed:

$$p(\text{true state } i|\mathbf{h}) = \frac{p(\mathbf{h}|\text{true state } i)q_i}{\sum_j p(\mathbf{h}|\text{true state } j)q_j}$$

Finally, the MAP estimator is given by $\hat{\mathbf{h}} = \mathbf{c}_{\text{argmax}_i\, p(\text{true state } i|\mathbf{h})}$. However, because learning requires that DOLCE's dynamics be differentiable, we use a soft version of MAP which involves using $\bar{\mathbf{h}} = \sum_i \mathbf{c}_i p(\text{true state } i|\mathbf{h})$ instead of $\hat{\mathbf{h}}$ and incorporating a "temperature" parameter into $\sigma_i$ as described below.

An important parameter in the mixture model is $T$, the number of true states (Gaussians bumps). Because $T$ directly corresponds to the number of states in the target FSM, if $T$ is chosen too small, DOLCE could not emulate the FSM. Consequently, we set $T$ to a large value, and the training procedure includes a technique for eliminating unnecessary true states. (If the initially selected $T$ is not large enough, the training procedure will not converge to zero error on the training set, and the procedure can be restarted with a larger value of $T$.)

At the start of training, each Gaussian center, $\mathbf{c}_i$, is initialized to a random location in the hidden state space. The standard deviations of each Gaussian, $\sigma_i$, are initially set to a large value. The priors, $q_i$, are set to $1/T$. The weights are set to initial values chosen from the uniform distribution in $[-.25,.25]$. All connection weights feeding into the hidden layer are second order.

The network weights and mixture model parameters—$\mathbf{C}$, $\sigma$, and $\mathbf{q}$—are adjusted by gradient descent in a cost measure, $\mathcal{C}$. This cost includes two components: (a) the performance error, $\mathcal{E}$, which is a squared difference between the actual and target network output following presentation of a training string, and (b) a complexity

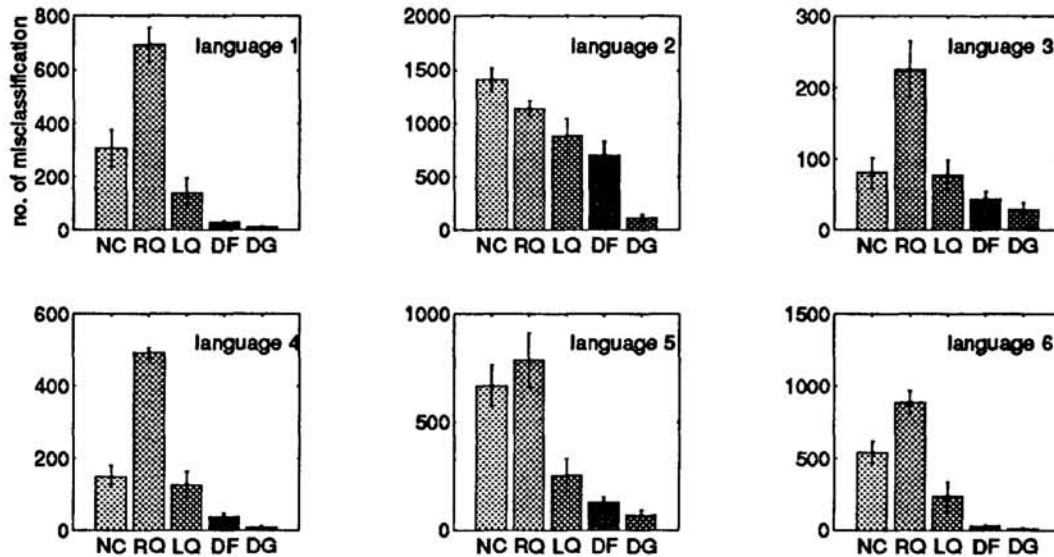

Figure 4: Each graph depicts generalization performance on one of the Tomita languages for 5 alternative neural net approaches: no clustering [NC], rigid quantization [RQ], learn then quantize [LQ], DOLCE in unsupervised mode using Forgy's algorithm [DF], DOLCE in supervised mode using mixture model [DG]. The vertical axis shows the number of misclassification of 3000 test strings. Each bar is the average result across 10 replications with different initial weights.

cost, which is the entropy of the prior distribution, $q$:

$$\mathcal{C} = \mathcal{E} - \lambda \sum q_i \ln q_i$$

where $\lambda$ is a regularization parameter. The complexity cost is minimal when only one Gaussian has a nonzero prior, and maximal when all priors are equal. Hence, the cost encourages unnecessary Gaussians to drop out of the mixture model.

The particular gradient descent procedure used is a generalization of back propagation through time (Rumelhart, Hinton, & Williams, 1986) that incorporates the mixture model. To better condition the search space and to avoid a constrained search, optimization is performed not over $\sigma$ and $q$ directly but rather over hyperparameters a and b, where $\sigma_i^2 = \exp(a_i)/\beta$ and $q_i = \exp(-b_i^2)/\sum_j \exp(-b_j^2)$.

The global parameter $\beta$ scales the overall spread of the Gaussians, which corresponds to the level of noise in the model. As performance on the training set improves, we assume that the network weights are coming to better approximate the target weights, hence that the level of noise is decreasing. Thus, we tie $\beta$ to the performance error $\mathcal{E}$. We have used various annealing schedules and DOLCE appears robust to this variation; we currently use $\beta \propto 1/\mathcal{E}$. Note that as $\mathcal{E} \to 0$, $\beta \to \infty$ and the probability density under one Gaussian at $h$ will become infinitely greater than the density under any other; consequently, the soft MAP estimator, $\bar{h}$, becomes equivalent to the MAP estimator $\hat{h}$, and the transformed hidden state becomes discrete. A schematic depiction of the probability landscape both before and after training is depicted in Figure 3.

## 5   SIMULATION STUDIES

The network was trained on a set of regular languages first studied by Tomita (1982). The languages, which utilize only the symbols 0 and 1, are: (1) $1^*$; (2) $(10)^*$; (3) no odd number of consecutive 1's is directly followed by an odd number of consecutive 0's; (4) any string not containing the substring "000"; (5) $[(01|10)(01|10)]^*$; (6) the difference between the number of ones and number of zeros in the string is a multiple of three; and (7) $0^*1^*0^*1^*$.

A fixed training corpus of strings was generated for each language, with an equal number of positive and negative examples. The maximum string length varied from 5 to 10 symbols and the total number of examples varied from 50 to 150, depending on the difficulty of the induction task.

Each string was presented one symbol at a time, after which DOLCE was given a target output that specified whether the string was a positive or negative example of the language. Training continued until DOLCE converged on a set of weights and mixture model parameters. Because we assume that the training examples are correctly classified, the error $\mathcal{E}$ on the training set should go to zero when DOLCE has learned. If this did not happen on a given training run, we restarted the simulation with different initial random weights.

For each language, ten replications of DOLCE (with the supervised mixture model) were trained, each with different random initial weights. The learning rate and regularization parameter $\lambda$ were chosen for each language by quick experimentation with the aim of maximizing the likelihood of convergence on the training set. We also trained a version of DOLCE that clustered using the unsupervised Forgy algorithm, as well as several alternative neural net approaches: a generic recurrent net, as shown on the left side of Figure 1, which used no clustering [NC]; a version with rigid quantization during training [RQ], comparable to the earlier work of Zeng, Goodman, and Smyth (1993); and a version in which the unsupervised Forgy algorithm was used to quantize the hidden state following training [LQ], comparable to the earlier work of Das and Das (1991). In these alternative approaches, we used the same architecture as DOLCE except for the clustering procedure. We selected learning parameters to optimize performance on the training set, ran ten replications for each language, replaced runs which did not converge, and used the same training sets.

## 6   RESULTS AND CONCLUSION

In Figure 4, we compare the generalization performance of DOLCE—both the unsupervised Forgy [DF] and supervised mixture model [DG]—to the NC, RQ, and LQ approaches. Generalization performance was tested using 3000 strings not in the training set, half positive examples and half negative. The two versions of DOLCE outperformed the alternative neural net approaches, and the DG version of DOLCE consistently outperformed the DF version.

To summarize, we have described an approach that incorporates inductive bias into a learning algorithm in order to encourage the evolution of discrete representations during training. This approach is a quite general and can be applied to domains

other than grammaticality judgement where discrete representations might be desirable. Also, this approach is not specific to recurrent networks and may be applied to feedforward networks. We are now in the process of applying DOLCE to a much larger, real-world problem that involves predicting the next symbol in a string. The data base comes from a case study in software engineering, where each symbol represents an operation in the software development process. This data is quite noisy and it is unlikely that the data can be parsimoniously described by an FSM. Nonetheless, our initial results are encouraging: DOLCE produces predictions at least three times more accurate than a standard recurrent net without clustering.

## Acknowledgements

This research was supported by NSF Presidential Young Investigator award IRI–9058450 and grant 90–21 from the James S. McDonnell Foundation.

## References

S. Das & R. Das. (1991) Induction of discrete state-machine by stabilizing a continuous recurrent network using clustering. *Computer Science and Informatics* 21(2):35-40. Special Issue on Neural Computing.

J.L. Elman. (1990) Finding structure in time. *Cognitive Science* 14:179-212.

E. Forgy. (1965) Cluster analysis of multivariate data: efficiency versus interpretability of classifications. *Biometrics* 21:768-780.

M.C. Mozer & J.D Bachrach. (1990) Discovering the structure of a reactive environment by exploration. *Neural Computation* 2(4):447-457.

C. McMillan, M.C. Mozer, & P. Smolensky. (1992) Rule induction through integrated symbolic and subsymbolic processing. In J.E. Moody, S.J. Hanson, & R.P. Lippmann (eds.), *Advances in Neural Information Processing Systems 4*, 969-976. San Mateo, CA: Morgan Kaufmann.

C.L. Giles, D. Chen, C.B. Miller, H.H. Chen, G.Z. Sun, & Y.C. Lee. (1992) Learning and extracting finite state automata with second-order recurrent neural network. *Neural Computation* 4(3):393-405.

H. Schütze. (1993) Word space. In S.J. Hanson, J.D. Cowan, & C.L. Giles (eds.), *Advances in Neural Information Processing Systems 5*, 895-902. San Mateo, CA: Morgan Kaufmann.

M. Tomita. (1982) Dynamic construction of finite-state automata from examples using hill-climbing. *Proceedings of the Fourth Annual Conference of the Cognitive Science Society*, 105-108.

G. Towell & J. Shavlik. (1992) Interpretion of artificial neural networks: mapping knowledge-based neural networks into rules. In J.E. Moody, S.J. Hanson, & R.P. Lippmann (eds.), *Advances in Neural Information Processing Systems 4*, 977-984. San Mateo, CA: Morgan Kaufmann.

R.L. Watrous & G.M. Kuhn. (1992) Induction of finite state languages using second-order recurrent networks. In J.E. Moody, S.J. Hanson, & R.P. Lippmann (eds.), *Advances in Neural Information Processing Systems 4*, 969-976. San Mateo, CA: Morgan Kaufmann.

Z. Zeng, R. Goodman, & P. Smyth. (1993) Learning finite state machines with self-clustering recurrent networks. *Neural Computation* 5(6):976-990.
